# MODELS OF OCULAR DOMINANCE COLUMN FORMATION: ANALYTICAL AND COMPUTATIONAL RESULTS

Kenneth D. Miller
UCSF Dept. of Physiology

Joseph B. Keller      SF, CA 94143-0444      Michael P. Stryker
Mathematics Dept., Stanford    ken@phyb.ucsf.edu    Physiology Dept., UCSF

## ABSTRACT

We have previously developed a simple mathematical model for formation of ocular dominance columns in mammalian visual cortex. The model provides a common framework in which a variety of activity-dependent biological machanisms can be studied. Analytic and computational results together now reveal the following: if inputs specific to each eye are locally correlated in their firing, and are not anticorrelated within an arbor radius, monocular cells will robustly form and be organized by intra-cortical interactions into columns. Broader correlations within each eye, or anti-correlations between the eyes, create a more purely monocular cortex; positive correlation over an arbor radius yields an almost perfectly monocular cortex. Most features of the model can be understood analytically through decomposition into eigenfunctions and linear stability analysis. This allows prediction of the widths of the columns and other features from measurable biological parameters.

## INTRODUCTION

In the developing visual system in many mammalian species, there is initially a uniform, overlapping innervation of layer 4 of the visual cortex by inputs representing the two eyes. Subsequently, these inputs segregate into patches or stripes that are largely or exclusively innervated by inputs serving a single eye, known as ocular dominance patches. The ocular dominance patches are on a small scale compared to the map of the visual world, so that the initially continuous map becomes two interdigitated maps, one representing each eye. These patches, together with the layers of cortex above and below layer 4, whose responses are dominated by the eye innervating the corresponding layer 4 patch, are known as ocular dominance columns.

The discoveries of this system of ocular dominance and many of the basic features of its development were made by Hubel and Wiesel in a series of pioneering studies in the 1960s and 1970s (e.g. Wiesel and Hubel (1965), Hubel, Wiesel and LeVay (1977)). A recent brief review is in Miller and Stryker (1989).

The segregation of patches depends on local correlations of neural activity that are very much greater within neighboring cells in each eye than between the two eyes. Forcing the eyes to fire synchronously prevents segregation, while forcing them to fire more asynchronously than normally causes a more complete segregation than normal. The segregation also depends on the activity of cortical cells. Normally, if one eye is closed in a young kitten during a critical period for developmental plasticity ("monocular deprivation"), the more active, open eye comes to dominantly or exclusively drive most cortical cells, and the inputs and influence of the closed eye become largely confined to small islands of cortex. However, when cortical cells are inhibited from firing, the opposite is the case: the less active eye becomes dominant, suggesting that it is the correlation between pre- and post-synaptic activation that is critical to synaptic strengthening.

We have developed and analyzed a simple mathematical model for formation of ocular dominance patches in mammalian visual cortex, which we briefly review here (Miller, Keller, and Stryker, 1986). The model provides a common framework in which a variety of activity-dependent biological models, including Hebb synapses and activity-dependent release and uptake of trophic factors, can be studied. The equations are similar to those developed by Linsker (1986) to study the development of orientation selectivity in visual cortex. We have now extended our analysis and also undertaken extensive simulations to achieve a more complete understanding of the model. Many results have appeared, or will appear, in more detail elsewhere (Miller, Keller and Stryker, 1989; Miller and Stryker, 1989; Miller, 1989).

## EQUATIONS

Consider inputs carrying information from two eyes and co-innervating a single cortical sheet. Let $S^L(x, \delta, t)$ and $S^R(x, \delta, t)$, respectively, be the left eye and right eye synaptic weight from eye-coordinate $\delta$ to cortical coordinate $x$ at time $t$. Consideration of simple activity-dependent models of synaptic plasticity, such as Hebb synapses or activity-dependent release and uptake of trophic or modification factors, leads to equations for the time development of $S^L$ and $S^R$:

$$\partial_t S^J(x, \delta, t) = \lambda A(x-\delta) \sum_{y,\beta,K} I(x-y) C^{JK}(\delta-\beta) S^K(y, \beta, t) - \gamma S^K(x, \delta, t) - \epsilon. \quad (1)$$

$J, K$ are variables which each may take on the values $L, R$. $A(x-\delta)$ is a connectivity function, giving the number of synapses from $\delta$ to $x$ (we assume an identity mapping from eye coordinates to cortical coordinates). $C^{JK}(\delta - \beta)$ measures the correlation in firing of inputs from eyes $J$ and $K$ when the inputs are separated by the distance $\delta - \beta$. $I(x - y)$ is a model-dependent spread of influence across cortex, telling how two synapses which fire synchronously, separated by the distance $x - y$, will influence

one another's growth. This influence incorporates lateral synaptic interconnections in the case of Hebb synapses (for linear activation, $\mathbf{I} = (\mathbf{1} - \mathbf{B})^{-1}$, where $\mathbf{1}$ is the identity matrix and $\mathbf{B}$ is the matrix of cortico-cortical synaptic weights), and incorporates the effects of diffusion of trophic or modification factors in models involving such factors. $\lambda, \gamma$ and $\epsilon$ are constants. Constraints to conserve or limit the total synaptic strength supported by a single cell, and nonlinearities to keep left- and right-eye synaptic weights positive and less than some maximum, are added.

Subtracting the equation for $S^R$ from that for $S^L$ yields a model equation for the difference, $S^D(x, \delta, t) \equiv S^L(x, \delta, t) - S^R(x, \delta, t)$:

$$\partial_t S^D(x, \delta, t) = \lambda A(x - \delta) \sum_{y, \beta} I(x - y) C^D(\delta - \beta) S^D(y, \beta, t) - \gamma S^D(x, \delta, t). \quad (2)$$

Here, $C^D = C^{\text{SameEye}} - C^{\text{OppEye}}$, where $C^{\text{SameEye}} = C^{LL} = C^{RR}$, $C^{\text{OppEye}} = C^{LR} = C^{RL}$, and we have assumed statistical equality of the two eyes.

## SIMULATIONS

The development of equation (1) was studied in simulations using three 25 x 25 grids for the two input layers, one representing each eye, and a single cortical layer. Each input cell connects to a $7 \times 7$ square arbor of cortical cells centered on the corresponding grid point ($A(x) = 1$ on the square of $\pm 3$ grid points, 0 otherwise). Initial synaptic weights are randomly assigned from a uniform distribution between 0.8 and 1.2. Synapses are allowed to decrease to 0 or increase to a weight of 8. Synaptic strength over each cortical cell is conserved by subtracting after each iteration from each active synapse the average change in synaptic strength on that cortical cell. Periodic boundary conditions on the three grids are used.

A typical time development of the cortical pattern of ocular dominance is shown in figure 1. For this simulation, correlations within each eye decrease with distance to zero over 4–5 grid points (circularly symmetric gaussian with parameter 2.8 grid points). There are no opposite-eye correlations. The cortical interaction function is a "Mexican hat" function excitatory between nearest neighbors and weakly inhibitory more distantly ($I(x) = \exp\left(\left(\frac{-|x|}{\lambda_I}\right)^2\right) - \frac{1}{9}\exp\left(\left(\frac{-|x|}{3\lambda_I}\right)^2\right)$, $\lambda_I = 0.93$.) Individual cortical cell receptive fields refine in size and become monocular (innervated exclusively by a single eye), while individual input arbors refine in size and become confined to alternating cortical stripes (not shown).

Dependence of these results on the correlation function is shown in figure 2. Wider ranging correlations within each eye, or addition of opposite-eye anticorrelations, increase the monocularity of cortex. Same-eye anticorrelations decrease monocularity, and if significant within an arbor radius (i.e. within $\pm 3$ grid points for the $7 \times 7$ square arbors) tend to destory the monocular organization, as seen at the lower right. Other simulations (not shown) indicate that same-eye correlation over nearest neighbors is sufficient to give the periodic organization of ocular dominance, while correlation over an arbor radius gives an essentially fully monocular cortex.

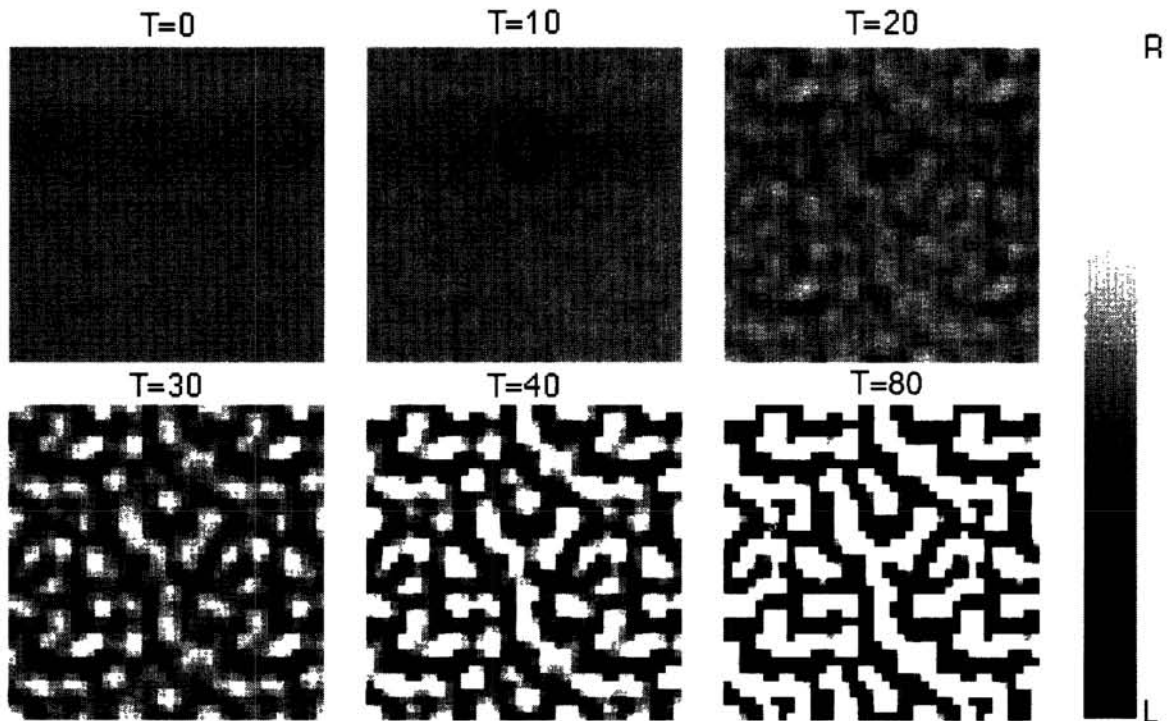

*Figure 1. Time development of cortical ocular dominance. Results shown after 0, 10, 20, 30, 40, 80 iterations. Each pixel represents ocular dominance $(\sum_\alpha S^D(x, \alpha))$ of a single cortical cell. $40 \times 40$ pixels are shown, repeating 15 columns and rows of the cortical grid, to reveal the pattern across the periodic boundary conditions.*

Simulation of time development with varying cortical interaction and arbor functions shows complete agreement with the analytical results presented below. The model also reproduces the experimental effects of monocular deprivation, including the presence of a critical developmental period for this effect.

## EIGENFUNCTION ANALYSIS

Consider an initial condition for which $S^D \approx 0$, and near which equation (2) linearizes some more complex, nonlinear biological reality. $S^D \equiv 0$ is a time-independent solution of equation (2). Is this solution stable to small perturbations, so that equality of the two eyes will be restored after perturbation, or is it unstable, so that a pattern of ocular dominance will grow? If it is unstable, which pattern will initially grow the fastest? These are inherently linear questions: they depend only on the behavior of the equations when $S^D$ is small, so that nonlinear terms are negligible.

To solve this problem, we find the characteristic, independently growing modes of equation (2). These are the eigenfunctions of the operator on the right side of that equation. Each mode grows exponentially with growth rate given by its eigenvalue. If any eigenvalue is positive (they are real), the corresponding mode will grow. Then the $S^D \equiv 0$ solution is unstable to perturbation.

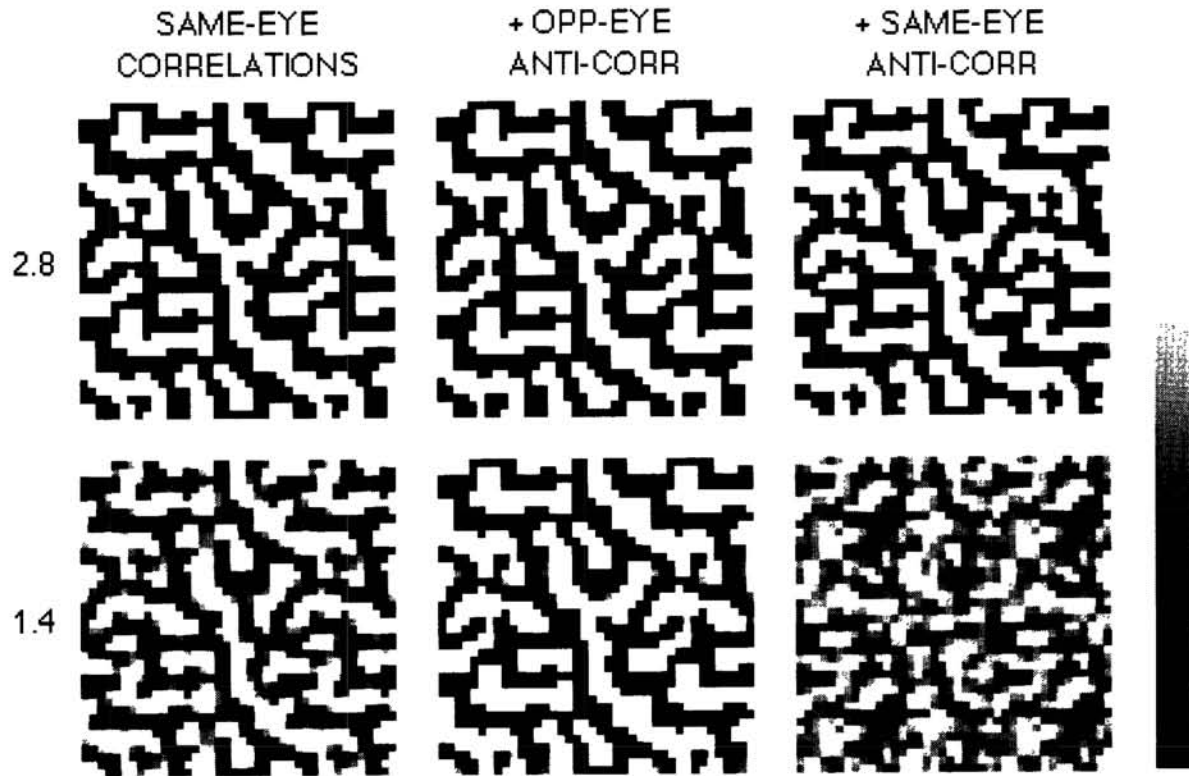

*Figure 2. Cortical ocular dominance after 200 iterations for 6 choices of correlation functions. Top left is simulation of figure 1. Top and bottom rows correspond to gaussian same-eye correlations with parameter 2.8 and 1.4 grid points, respectively. Middle column shows the effect of adding weak, broadly ranging anticorrelations between the two eyes (gaussian with parameter 3 times larger than, and amplitude $-\frac{1}{9}$ that of, the same-eye correlations). Right column shows the effect of instead adding the anticorrelation to the same-eye correlation function.*

## ANALYTICAL CHARACTERIZATION OF EIGENFUNCTIONS

Change variables in equation (2) from cortex and inputs, $(x, \alpha)$, to cortex and receptive field, $(x, r)$ with $r \equiv x - \alpha$. Then equation 2 becomes a convolution in the cortical variable. The result (assume a continuum; results on a grid are similar) is that eigenfunctions are of the form $S^D_{m,\xi}(x, \alpha, t) = e^{im \cdot x} RF_{m,\xi}(r)$. $RF_{m,\xi}$ is a characteristic receptive field, representing the variation of the eigenfunction as $r$ varies while cortical location $x$ is fixed. $m$ is a pair of real numbers specifying a two dimensional wavenumber of cortical oscillation, and $\xi$ is an additional index enumerating $RF$s for a given $m$. The eigenfunctions can also be written $e^{im \cdot \alpha} ARB_{m\gamma}(r)$ where $ARB_{m\gamma}(r) = e^{im \cdot r} RF_{m\gamma}(r)$. $ARB$ is a characteristic arbor, representing the variation of the eigenfunction as $r$ varies while input location $\alpha$ is fixed. While these functions are complex, one can construct real eigenfunctions from them with similar properties (Miller and Stryker, 1989). A monocular (real) eigenfunction is illustrated in figure 3.

## CHARACTERISTIC RECEPTIVE FIELD

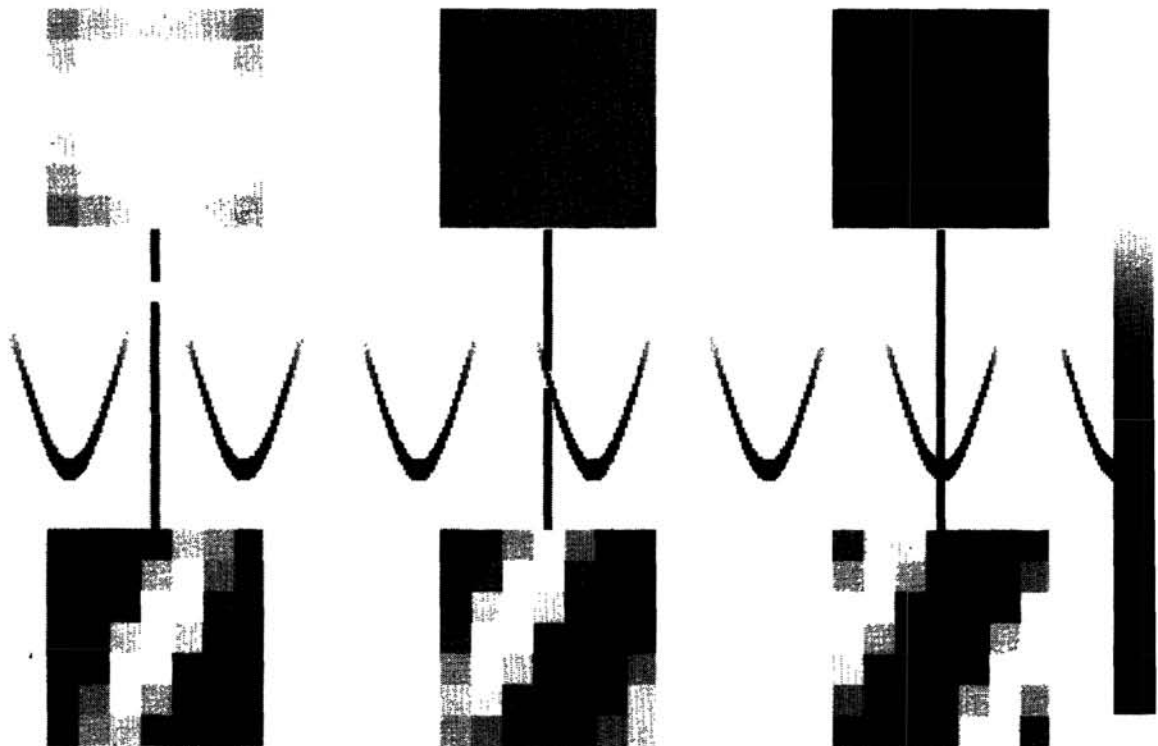

## CHARACTERISTIC ARBOR

*Figure 3. One of the set (identical but for rotations and reflections) of fastest-growing eigenfunctions for the functions used in figure 1. The monocular receptive fields of synaptic differences $S^D$ at different cortical locations, the oscillation across cortex, and the corresponding arbors are illustrated.*

Modes with $RF$s dominated by one eye ($\sum_y RF_{m,\xi}(y) \neq 0$) will oscillate in dominance with wavelength $\frac{2\pi}{m}$ across cortex. A monocular mode is one for which $RF$ does not change sign. The oscillation of monocular fields, between domination by one eye and domination by the other, yields ocular dominance columns. The fastest growing mode in the linear regime will dominate the final pattern: if its receptive field is monocular, its wavelength will determine the width of the final columns.

One can characterize the eigenfunctions analytically in various limiting cases. The general conclusion is as follows. The fastest growing mode's receptive field $RF$ is largely determined by the correlation function $C^D$. If the peak of the fourier transform of $C^D$ corresponds to a wavelength much larger than an arbor diameter, the mode will be monocular; if it corresponds to a wavelength smaller than an arbor diameter, the mode will be binocular. If $C^D$ selects a monocular mode, a broader $C^D$ (more sharply peaked fourier spectrum about wavenumber 0) will increase the dominance in growth rate of the monocular mode over other modes; in the limit

in which $C^D$ is constant with distance, only the monocular modes grow and all other modes decay. If the mode is monocular, the peak of the fourier transform of the cortical interaction function selects the wavelength of the cortical oscillation, and thus selects the wavelength of ocular dominance organization. In the limit in which correlations are broad with respect to an arbor, one can calculate that the growth rate of monocular modes as a function of wavenumber of oscillation $m$ is proportional to $\sum_l \tilde{I}(m-l)\tilde{C}(l)\tilde{A}^2(l)$ (where $\tilde{X}$ is the fourier transform of $X$). In this limit, only $l$'s which are close to 0 can contribute to the sum, so the peak will occur at or near the $m$ which maximizes $\tilde{I}(m)$.

There is an exception to the above results if constraints conserve, or limit the change in, the total synaptic strength over the arbor of an input cell. Then monocular modes with wavelength longer than an arbor diameter are suppressed in growth rate, since individual inputs would have to gain or lose strength throughout their arborization. Given a correlation function that leads to monocular cells, a purely excitatory cortical interaction function would lead a single eye to take over all of cortex; however, if constraints conserve synaptic strength over an input arbor, the wavelength will instead be about an arbor diameter, the largest wavelength whose growth rate is not suppressed. Thus, ocular dominance segregation can occur with a purely excitatory cortical interaction function, though this is a less robust phenomenon. Analytically, a constraint conserving strength over afferent arbors, implemented by subtracting the average change in strength over an arbor at each iteration from each synapse in the arbor, transforms the previous expression for the growth rates to $\sum_l \tilde{I}(m-l)\tilde{C}(l)\tilde{A}^2(l)\left(1 - \frac{\tilde{A}(m)\tilde{A}(l)}{\tilde{A}^2(0)}\right)$.

## COMPUTATION OF EIGENFUNCTIONS

Eigenfunctions are computed on a grid, and the resulting growthrates as a function of wavelength are compared to the analytical expression above, in the absence of constraints on afferents. The results, for the parameters used in figure (2), are shown in figure (4). The grey level indicates monocularity of the modes, defined as $\sum_r RF(r)$ normalized on a scale between 0 and 1 (described in Miller and Stryker (1989)). The analytical expression for the growth rate, whose peak coincides in every case with the peak of $\tilde{I}(m)$, accurately predicts the growth rate of monocular modes, even far from the limiting case in which the expression was derived. Broader correlations or opposite-eye anticorrelations enhance the monocularity of modes and the growth rate of monocular modes, while same-eye anticorrelations have the opposite effects. When same-eye anticorrelations are short range compared to an arbor radius, the fastest growing modes are binocular.

Results obtained for calculations in the presence of constraints on afferents are also as predicted. With an excitatory cortical interaction function, the spectrum is radically changed by constraints, selecting a mode with a wavelength equal to an arbor diameter rather than one with a wavelength as wide as cortex. With the Mexican hat cortical interaction function used in the simulations, the constraints suppress the growth of long-wavelength monocular modes but do not alter the basic

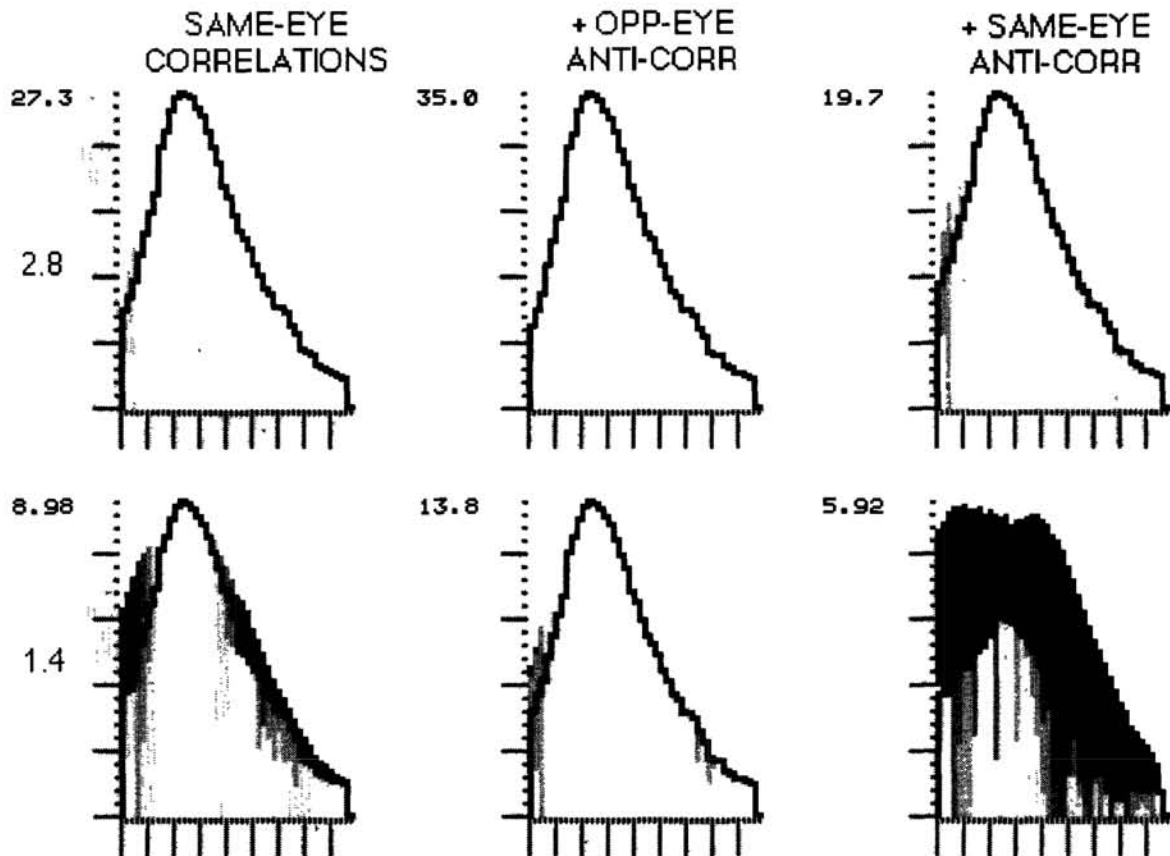

*Figure 4. Growth rate (vertical axis) as a function of inverse wavelength (horizontal axis) for the six sets of functions used in figure 2, computed on the same grids. Grey level codes maximum monocularity of modes with the given wavelength and growth rate, from fully monocular (    white) to fully binocular (black). The black curve indicates the prediction for relative growth rates of monocular modes given in the limit of broad correlations, as described in the text.*

structure or peak of the spectrum.

## CONNECTIONS TO OTHER MODELS

The model of Swindale (1980) for ocular dominance segregation emerges as a limiting case of this model when correlations are constant over a bit more than an arbor diameter. Swindale's model assumed an effective interaction between synapses depending only on eye of origin and distance across cortex. Our model gives a biological underpinning to this effective interaction in the limiting case, allows consideration of more general correlation functions, and allows examination of the development of individual arbors and receptive fields and their relationships as well as of overall ocular dominance.

Equation 2 is very similar to equations studied by others (Linsker, 1986, 1988; Sanger, this volume). There are several important differences in our results. First, in this model synapses are constrained to remain positive. Biological synapses are

either exclusively positive or exclusively negative, and in particular the projection of visual input to visual cortex is purely excitatory. Even if one is modelling a system in which there are both excitatory and inhibitory inputs, these two populations will almost certainly be statistically distinct in their activities and hence not treatable as a single population whose strengths may be either positive or negative. $S^D$, on the other hand, is a biological variable which starts near 0 and may be either positive or negative. This allows for a linear analysis whose results will remain accurate in the presence of nonlinearities, which is crucial for biology.

Second, we analyze the effect of intracortical synaptic interactions. These have two impacts on the modes: first, they introduce a phase variation or oscillation across cortex. Second, they typically enhance the growth rate of monocular modes relative to modes whose sign varies across the receptive field.

## Acknowledgements

Supported by an NSF predoctoral fellowship and by grants from the McKnight Foundation and the System Development Foundation. Simulations were performed at the San Diego Supercomputer Center.

## References

Hubel, D.H., T.N. Wiesel and S. LeVay, 1977. Plasticity of ocular dominance columns in monkey striate cortex, Phil. Trans. R. Soc. Lond. B. **278**:377-409.

Linsker, R., 1986. From basic network principles to neural architecture, *Proc. Natl. Acad. Sci. USA* **83**:7508-7512, 8390-8394, 8779-8783.

Linsker, R., 1988. Self-Organization in a Perceptual Network. *IEEE Computer* **21**:105-117.

Miller, K.D., 1989. Correlation-based models of neural development, to appear in **Neuroscience and Connectionist Theory** (M.A. Gluck & D.E. Rumelhart, Eds.), Hillsdale, NJ: Lawrence Erlbaum Associates.

Miller, K.D., J.B. Keller & M.P. Stryker, 1986. Models for the formation of ocular dominance columns solved by linear stability analysis, *Soc. Neurosc. Abst.* **12**:1373.

Miller, K.D., J.B. Keller & M.P. Stryker, 1989. Ocular dominance column development: analysis and simulation. Submitted for publication.

Miller, K.D. & M.P. Stryker, 1989. The development of ocular dominance columns: mechanisms and models, to appear in **Connectionist Modeling and Brain Function: The Developing Interface** (S. J. Hanson & C. R. Olson, Eds.), MIT Press/ Bradford.

Sanger, T.D., 1989. An optimality principle for unsupervised learning, this volume.

Swindale, N.V., 1980. A model for the formation of ocular dominance stripes, *Proc. R. Soc. Lond. B.* **208**:265-307.

Wiesel, T.N. & D.H. Hubel, 1965. Comparison of the effects of unilateral and bilateral eye closure on cortical unit responses in kittens, J. Neurophysiol. **28**:, 1029-1040.